# THEORY OF SELF-ORGANIZATION OF CORTICAL MAPS

Shigeru Tanaka
Fundamental Research Laboratorys, NEC Corporation
1-1 Miyazaki 4-Chome, Miyamae-ku, Kawasaki, Kanagawa 213, Japan

## ABSTRACT

We have mathematically shown that cortical maps in the primary sensory cortices can be reproduced by using three hypotheses which have physiological basis and meaning. Here, our main focus is on ocular dominance column formation in the primary visual cortex. Monte Carlo simulations on the segregation of ipsilateral and contralateral afferent terminals are carried out. Based on these, we show that almost all the physiological experimental results concerning the ocular dominance patterns of cats and monkeys reared under normal or various abnormal visual conditions can be explained from a viewpoint of the phase transition phenomena.

## ROUGH SKETCH OF OUR THEORY

In order to describe the use-dependent self-organization of neural connections {Singer,1987 and Frank,1987}, we have proposed a set of coupled equations involving the electrical activities and neural connection density {Tanaka, 1988}, by using the following physiologically based hypotheses: (1) Modifiable synapses grow or collapse due to the competition among themselves for some trophic factors, which are secreted retrogradely from the postsynaptic side to the presynaptic side. (2) Synapses also sprout or retract according to the concurrence of presynaptic spike activity and postsynaptic local membrane depolarization. (3) There already exist lateral connections within the layer, into which the modifiable nerve fibers are destined to project, before the synaptic modification begins. Considering this set of equations, we find that the time scale of electrical activities is much smaller than time course necessary for synapses to grow or retract. So we can apply the adiabatic approximation to the equations. Furthermore, we identify the input electrical activities, i.e., the firing frequency elicited from neurons in the projecting neuronal layer, with the stochastic process which is specialized by the spatial correlation function $C_{k\mu;k'\mu'}$. Here, k and k' represent the positions of the neurons in the projecting layer. $\mu$ stands for different pathways such as ipsilateral or contralateral, on-center or off-center, colour specific or nonspecific and so on. From these approximations, we have a nonlinear

stochastic differential equation for the connection density, which describes a survival process of synapses within a small region, due to the strong competition. Therefore, we can look upon an equilibrium solution of this equation as a set of the Potts spin variables $\sigma_{jk\mu}$'s {Wu, 1982}. Here, if the neuron k in the projecting layer sends the axon to the position j in the target layer, $\sigma_{jk\mu}=1$ and if not, $\sigma_{jk\mu}=0$. The Potts spin variable has the following property:

$$\sum_{k\mu} \sigma_{jk\mu} = 1 \quad .$$

If we limit the discussion within such equilibrium solutions, the problem is reduced to the thermodynamics in the spin system. The details of the mathematics are not argued here because they are beyond the scope of this paper {Tanaka}. We find that equilibrium behavior of the modifiable nerve terminals can be described in terms of thermodynamics in the system in which Hamiltonian H and fictitious temperature T are given by

$$H = -q \sum_{jk\mu} \eta_{k\mu} \sigma_{jk\mu} - \sum_{jk\mu} \sum_{j'k'\mu'} V_{jj'} C_{k\mu;\,k'\mu'} \sigma_{jk\mu} \sigma_{j'k'\mu'} \quad , \tag{1}$$

$$T \propto \left( \frac{\tau_c}{\tau_s} \right)^{1/2} \quad , \tag{2}$$

where k and $C_{k\mu;\,k'\mu'}$ are the averaged firing frequency and the correlation function, respectively. $V_{jj'}$ describes interaction between synapses in the target layer. q is the ratio of the total averaged membrane potential to the averaged membrane potential induced through the modifiable synapses from the projecting layer. $\tau_c$ and $\tau_s$ are the correlation time of the electrical activities and the time course necessary for synapses to grow or collapse.

## APPLICATION TO THE OCULAR DOMINANCE COLUMN FORMATION

A specific cortical map structure is determined by the choice of the correlation function and the synaptic interaction function. Now, let us neglect k dependence of the correlation function and take into account only ipsilateral and contralateral pathways denoted by μ, for mathematical simplicity. In this case, we can reduce the Potts spin variable into the Ising spin one through the following transformation:

$$S_j = \sum_{k\mu} \mu \, \sigma_{jk\mu} \quad ,$$

where $j$ is the position in the layer 4 of the primary visual cortex, and $s_j$ takes only $+1$ or $-1$, according to the ipsilateral or contralateral dominance. We find that this system can be described by Hamiltonian:

$$H = -h\sum_j S_j - \frac{J}{2}\sum_j \sum_{j' \neq j} V_{jj'} S_j S_{j'} \quad . \tag{3}$$

The first term of eq.(3) reflects the ocular dominance shift, while the second term is essential to the ocular dominance stripe segregation.

Here, we adopt the following simplified function as $V_{jj'}$:

$$V_{jj'} = \frac{q_{ex}}{\pi \lambda_{ex}^2} \Theta\,(\lambda_{ex} - d_{jj'}) - \frac{q_{inh}}{\pi \lambda_{inh}^2} \Theta\,(\lambda_{inh} - d_{jj'}) \quad , \tag{4}$$

where $d_{jj'}$ is the distance between $j$ and $j'$. $\lambda_{ex}$ and $\lambda_{inh}$ are determined by the extent of excitatory and inhibitory lateral connections, respectively. $\Theta$ is the step function. $q_{ex}$ and $q_{inh}$ are propotional to the membrane potentials induced by excitatory and inhibitory neurons {Tanaka}. It is not essential to the qualitative discussion whether the interaction function is given by the use of the step function, the Gaussian function, or others.

Next, we define $\overline{\eta}_{+1}$ and $\overline{\eta}_{-1}$ as the average firing frequencies of ipsilateral and contralateral retinal ganglion cells (RGCs), and $\xi_{\pm 1}^V$ and $\xi_{\pm 1}^S$ as their fluctuations which originate in the visually stimulated and the spontaneous firings of RGCs, respectively. These are used to calculate two new parameters, $r$ and $a$:

$$r = \frac{\overline{\xi_{+1}^V \xi_{-1}^V}}{\sqrt{\overline{(\xi_{+1}^V)^2 + (\xi_{+1}^S)^2}}\,\sqrt{\overline{(\xi_{-1}^V)^2 + (\xi_{-1}^S)^2}}} \quad , \tag{5}$$

$$a = \frac{\overline{\eta}_{+1} - \overline{\eta}_{-1}}{\overline{\eta}_{+1} + \overline{\eta}_{-1}} \quad . \tag{6}$$

r is related to the correlation of firings elicited from the left and right RGCs. If there are only spontaneous firings, there is no correlation between the left and right RGCs' firings. On the other hand, in the presence of visual stimulation, they will correlate, since the two eyes receive almost the same images in normal animals. $a$ is a function of the imbalance of firings of the left and right RGCs. Now, J and h in eq.(3) can be expressed in terms of r and $a$:

$$J = b_1\left(1 - r\frac{1-a^2}{1+a^2}\right) \ ,$$

(7)

$$h = b_2 a \ ,$$

(8)

where $b_1$ is a constant of the order of 1, and $b_2$ is determined by average membrane potentials.

Using the above equations, it will now be shown that patterns such as the ones observed for the ocular dominance column of new-world monkeys and cats can be explained. The patterns are very much dependent on three parameters r, $a$ and $\kappa$ which is the ratio of the membrane potentials ($q_{inh}/q_{ex}$) induced by the inhibitory and excitatory neurons.

## RESULTS AND DISCUSSIONS

In the subsequent analysis by Monte Carlo simulations, we fix the values of parameters: $q_{ex} = 1.0$, $\lambda_{ex} = 0.25$, $\lambda_{inh} = 1.0$, $T = 0.25$, $b_1 = 1.0$, $b_2 = 0.1$, and $dx = 0.1$. $dx$ is the diameter of a small area which is occupied by one spin. In the computer simulations of Fig.1, we can see that the stripe patterns become more segregated as the correlation strength r decreases. The similarity of the pattern in Fig.1c to the well-known experimental evidence {Hubel and Wiesel, 1977} is striking. Furthermore, it is known that if the animal has been reared under the condition where the two optic nerves are electrically stimulated synchronously, stripes in the primary visual cortex are not formed {Stryker}. This condition corresponds to r values close to 1 and again our theory predicts these experimental results as can be seen in Fig.1a. On the contrary, if the strabismic animal has been reared under the normal condition {Wiesel and Hubel, 1974}, r is effectively smaller than that of a normal animal. So we expect that the ocular dominance stripe has very sharp delimitations as it is observed experimentally. In the case of a binocularly deprived animal {Wiesel and Hubel, 1974},i.e., $\xi_{+1}{}^v = \xi_{-1}{}^v = 0$, it is reasonable to expect that the situation is similar to the strabismic animal.

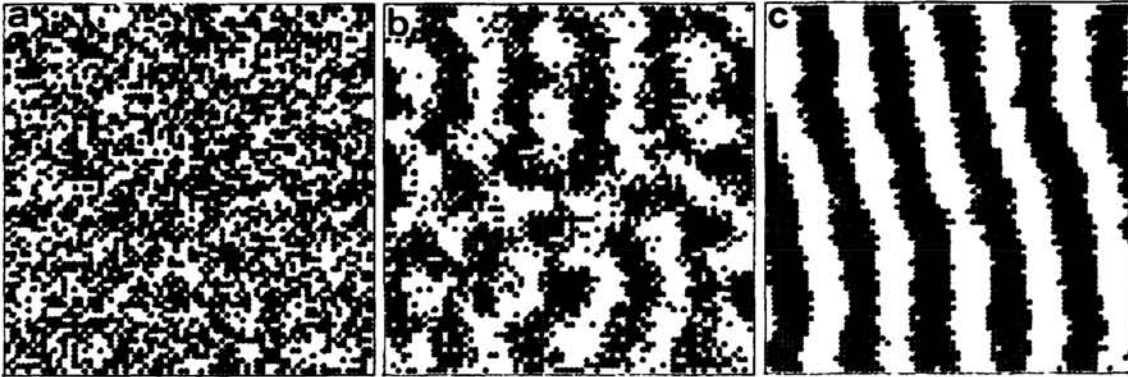

Figure 1. Ocular dominance patterns given by the computer simulations in the case of the large inhibitory connections ($\kappa = 1.0$) and the balanced activities ($\alpha = 0$). The correlation strength r is given in each case: $r = 0.9$ for (a), $r = 0.6$ for (b), and $r = 0.1$ for (c).

In the case of $\alpha \neq 0$, we can get asymmetric stripe patterns such as one in Fig.2a. Since this situation corresponds to the condition of the monocular deprivation, we can also explain the experimental observation {Hubel et al.,1977} successfully. There are other patterns seen in Fig.2b, which we call blob lattice patterns. The existence of such patterns has not been confirmed physiologically, as far as we know. However, this theory on the ocular dominance column formation predicts that the blob lattice patterns will be found if appropriate conditions, such as the period of the monocular

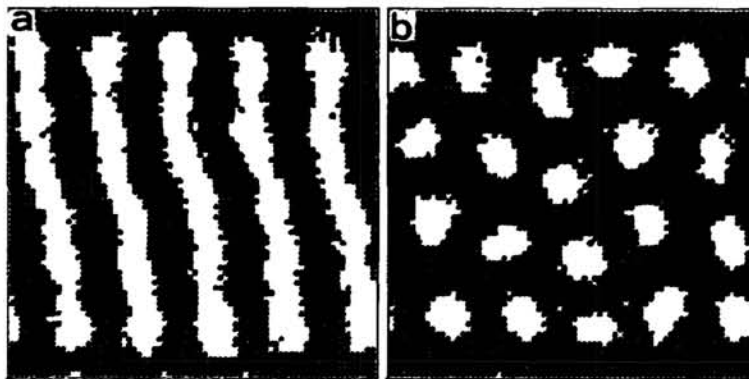

Figure 2. Ocular dominance patterns given by the computer simulations in the case of the large inhibitory connections ($\kappa = 1.0$) and the imbalanced activities: $a = 0.2$ for (a) and $a = 0.4$ for (b). The correlation strength r is given by $r = 0.1$ for both (a) and (b).

deprivation, are chosen.

We find that the straightness of the stripe pattern is controlled by the parameter κ. Namely, if κ is large, i.e. inhibitory connections are more effective than excitatory ones, the pattern is straight. However if κ is small the pattern has many branches and ends. This is illustrated in Fig. 3c. We can get a pattern similar to the ocular dominance pattern of normal cats {Anderson et al., 1988}, if κ is small and $r \simeq r_c$ (Fig.3b). The meaning of $r_c$ will be discussed in the following paragraphs. We further get a labyrinth pattern by means of r smaller than $r_c$ and the same κ. We can think κ value is specific to the animal under consideration because of its definition. Therefore, this theory also predicts that the ocular dominance pattern of the strabismic cat will be sharply delimitated but not a straight stripe in contrast to the pattern of monkey.

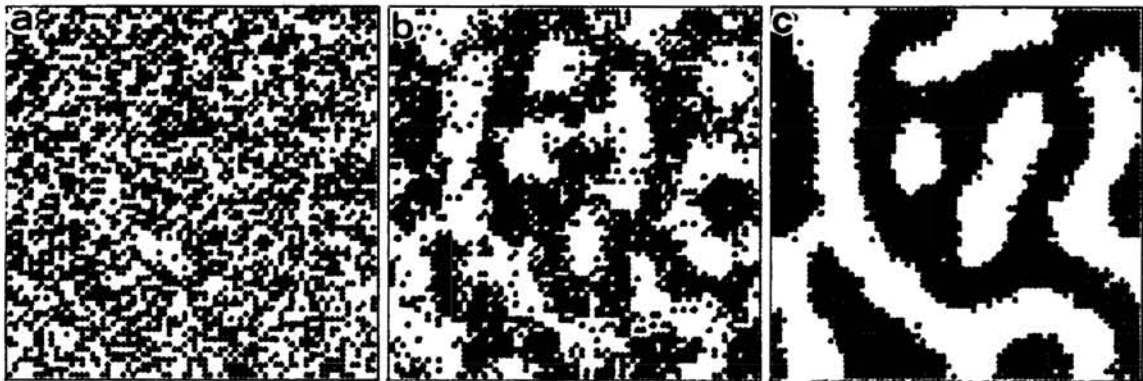

Figure 3. Ocular dominance patterns given by the computer simulations in the case of the small inhibitory connections (κ=0.3) and the balanced activities($a$=0). The correlation strength r is given in each case: r=0.9 for (a), r=0.6 for (b) and r=0.1 for (c).

Having seen specific examples, let us now discuss the importance of parameters r and $a$, which stand for the correlation strength and the imbalance of firings. According to qualitative difference of patterns obtained from our simulations, we classify the parameter space (r, $a$) into three regions in Fig.4: In region (S), stripe patterns appear. The left-eye dominance and the right-eye dominance bands are equal in width, for $a$=0. On the other hand, they are not equal for non-zero value. In region (B), patterns are blob lattices. In region (U), the patterns are uniform and we do not see any spatial modulation. A uniform pattern whose $a$ value is close to 0 is a random pattern, while if $a$ is close to 1 or −1 either ipsilateral or contralateral nerve terminals are present. On the horizontal axis, (S) and (U) regions are devided by the critical point $r_c$. In practice if we define the order parameter as the

ensemble-averaged amplitude of the dominant Fourier component of spatial patterns, and the susceptibility as the variance of the amplitude, then we can observe their singular behavior near $r = r_c$.

Various conditions where animals have been reared correspond positions in the parameter space of Fig.4: normal (N), synchronized electrical stimulation (SES), strabismus (S), binocular deprivation (BD), long-term monocular deprivation (LMD) and short-term monocular deprivation (SMD). If an animal is kept under the monocular deprivation for a long period, the absolute value of is close to 1 and r value is 0, considering eqs.(5) and (6). For a short-term monocular deprivation, the corresponding point falls on anywhere on the line from N to LMD, because relaxation from the symmetric stripe pattern to the open-eye dominant uniform pattern is incomplete. The position on this line is, therefore, determined by this relaxation period, in which the animal is kept under the monocular deprivation.

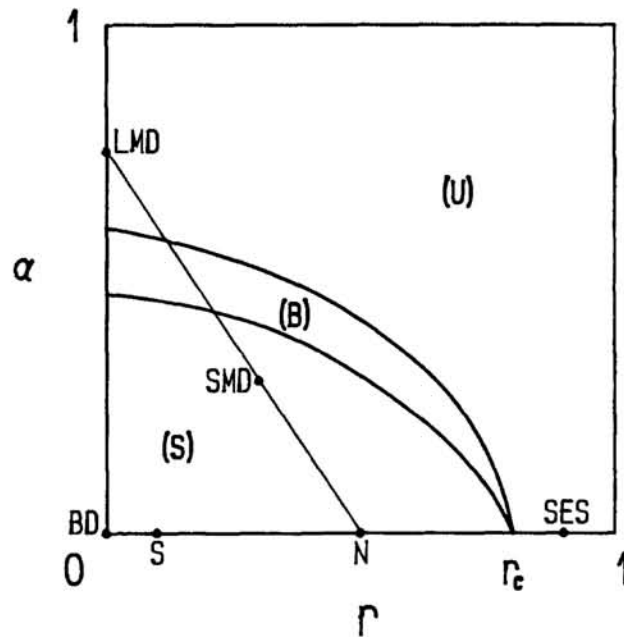

Figure 4. Schematic phase diagram for the pattern of ocular dominance columns. The parameter space (r, $a$) is devided into three regions: (S) stripe region, (B) blob lattice region, and (U) uniform region. N, SES, S, BD, LMD, and SMD stand for conditions: normal, synchronized electrical stimulation, strabismus, binocular deprivation, long-term monocular deprivation, and short-term monocular deprivation, respectively. We show only the diagram on the upper half plane, because the diagram is symmetrical with respect to the line of $a = 0$.

## CONCLUSION

In this report, a new theory has been proposed which is able to explain such use-dependent self-organization as the ocular dominance column formation. We have compared the theoretical results with various experimental data and excellent agreement is observed. We can also explain and predict self-organizing process of other cortical map structures such as the orientation column, the retinotopic organization, and so on. Furthermore, the three main hypotheses of this theory are not confined to the primary visual cortex. This suggests that the theory will have a wide applicability to the formation of cortical map structures seen in the somatosensory cortex {Kaas et al.,1983}, the auditory cortex {Knudsen et al.,1987}, and the cerebellum {Ito,1984}.

### References

P.A.Anderson, J.Olavarria, R.C.Van Sluyter, J.Neurosci. 8, 2184 (1988).
E.Frank, Trends in Neurosci. 10,188 (1987).
D.H.Hubel and T.N.Wiesel, Proc.R.Soc.Lond.B198,1(1977).
D.H.Hubel, T.N.Wiesel, S.LeVay, Phil.Trans.R.Soc. Lond. B278, 131 (1977).
M.Ito, The Cerebellum and Neural Control (Raven Press, 1984).
J.H.Kaas, M.M.Merzenich, H.P.Killackey, Ann. Rev. Neurosci. 6, 325 (1983).
E.I.Knudsen, S.DuLac, S.D.Esterly, Ann. Rev. Neurosci. 10, 41 (1987).
W.Singer,in The Neural and Molecular Bases of Learning (Hohn Wiley & Sons Ltd.,1987) pp.301-336;
M.P.Stryker, in Developmental Neurophysiology (Johns Hopkins Press), in press.
S.Tanaka, The Proceeding of SICE'88, ESS2-5, p. 1069 (1988).
S.Tanaka, to be submitted.
T.N.Wiesel and D.H.Hubel, J.Comp.Neurol.158, 307 (1974).
F.Y.Wu, Rev. Mod. Phys. 54, 235 (1982).
